# Design and Implementation of a High Speed CMAC Neural Network Using Programmable CMOS Logic Cell Arrays

W. Thomas Miller, III, Brian A. Box, and Erich C. Whitney
Department of Electrical and Computer Engineering
Kingsbury Hall
University of New Hampshire
Durham, New Hampshire 03824

James M. Glynn
Shenandoah Systems Company
1A Newington Park
West Park Drive
Newington, New Hampshire 03801

## Abstract

A high speed implementation of the CMAC neural network was designed using dedicated CMOS logic. This technology was then used to implement two general purpose CMAC associative memory boards for the VME bus. Each board implements up to 8 independent CMAC networks with a total of one million adjustable weights. Each CMAC network can be configured to have from 1 to 512 integer inputs and from 1 to 8 integer outputs. Response times for typical CMAC networks are well below 1 millisecond, making the networks sufficiently fast for most robot control problems, and many pattern recognition and signal processing problems.

## 1   INTRODUCTION

We have been investigating learning techniques for the control of robotic manipulators which utilize extensions of the CMAC neural network as developed by Albus

(1972; 1975; 1979). The learning control techniques proposed have been studied in our laboratory in a series of real time experimental studies (Miller, 1986; 1987; 1989; Miller et al., 1987; 1988; 1990). These studies successfully demonstrated the ability to learn the kinematics of a robot/video camera system interacting with randomly oriented objects on a moving conveyor, and to learn the dynamics of a multi-axis industrial robot during high speed motions. We have also investigated the use of CMAC networks for pattern recognition (Glanz and Miller, 1987; Herold et al., 1988) and signal processing (Glanz and Miller, 1989) applications, with encouraging results. The primary goal of this project was to implement a compact, high speed version of the CMAC neural network using CMOS logic cell arrays. Two prototype CMAC associative memory systems for the industry standard VME bus were then constructed.

## 2   THE CMAC NEURAL NETWORK

Figure 1 shows a simple example of a CMAC network with two inputs and one output. Each variable in the input state vector is fed to a series of input sensors with overlapping receptive fields. The width of the receptive field of each sensor produces input generalization, while the offset of the adjacent fields produces input quantization. The binary outputs of the input sensors are combined in a series of threshold logic units (called state space detectors) with thresholds adjusted to produce logical AND functions. Each of these units receives one input from the group of sensors for each input variable, and thus its input receptive field is the interior of a hypercube in the input hyperspace. The input sensors are interconnected in a sparse and regular fashion, so that each input vector excites a fixed number of state space detectors. The outputs of the state space detectors are connected randomly to a smaller set of threshold logic units (called multiple field detectors) with thresholds adjusted such that the output will be on if any input is on. The receptive field of each of these units is thus the union of the fields of many of the state space detectors. Finally, the output of each multiple field detector is connected, through an adjustable weight, to an output summing unit. The output for a given input is thus the sum of the weights selected by the excited multiple field detectors.

The nonlinear nature of the CMAC network is embodied in the interconnections of the input sensors, state space detectors, and multiple field detectors, which perform a fixed nonlinear associative mapping of the continuous valued input vector to a many dimensional binary valued vector (which has tens or hundreds of thousands of dimensions in typical implementations). The adaptation problem is linear in this many dimensional space, and all of the convergence theorems for linear adaptive elements apply.

## 3   THE CMAC HARDWARE DESIGN

The custom implementation of the CMAC associative memory required the development of two devices. The first device performs the input associative mapping, converting application relevant input vectors into traditional RAM addresses. The second device performs CMAC response accumulation, summing the weights from all excited receptive fields. Both devices were implemented using 70 MHz XILINX

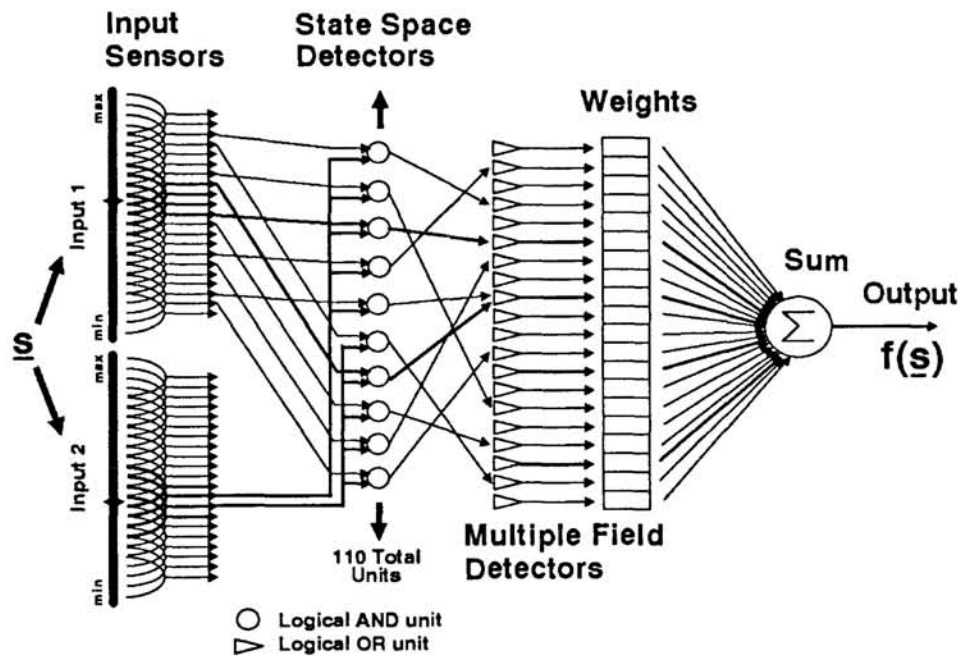

Figure 1: A Simple Example of a CMAC Neural Network

3090 programmable logic cell arrays.

The associative mapping device uses a bit recursive mapping scheme developed at UNH, which is similar in philosophy to the CMAC mapping proposed by Albus, but is structured for efficient implementation using discrete logic. The "address" of each excited virtual receptive field is formed recursively by clocking the input vector components sequentially from a buffer FIFO. The hashing of the virtual receptive field address to a physical RAM address is performed simultaneously, using pipelined logic. The resulting associative mapping generates one 18 bit RAM address for a given input vector. The multiple addresses, corresponding to the multiple receptive fields excited by a single input vector could be generated simultaneously using parallel addressing circuits, or sequentially using a single circuit.

The second CMAC device serves basically as an accumulator during CMAC response generation. As successive addresses are produced by the associative mapping circuit, the accumulator sums the corresponding values from the data RAM. During memory training, the response accumulation circuit adds the training adjustment to each of the addressed memory locations, placing the result back in the RAM. Eight independent CMAC output channels were placed on a single device.

In the final VME system design (Figure 2), a single CMAC associative mapping device was used. Overlapping receptive fields were implemented sequentially using the same device. A single CMAC response accumulation device was used, providing eight parallel output channels. A weight vector memory containing 1 million 8 bit weights was provided using 85 nanosecond 512 KByte static RAM SIMMs. A TMS320E15 microcontroller was utilized to supervise communications with the VME bus. The operational firmware for the microcontroller chip was designed to

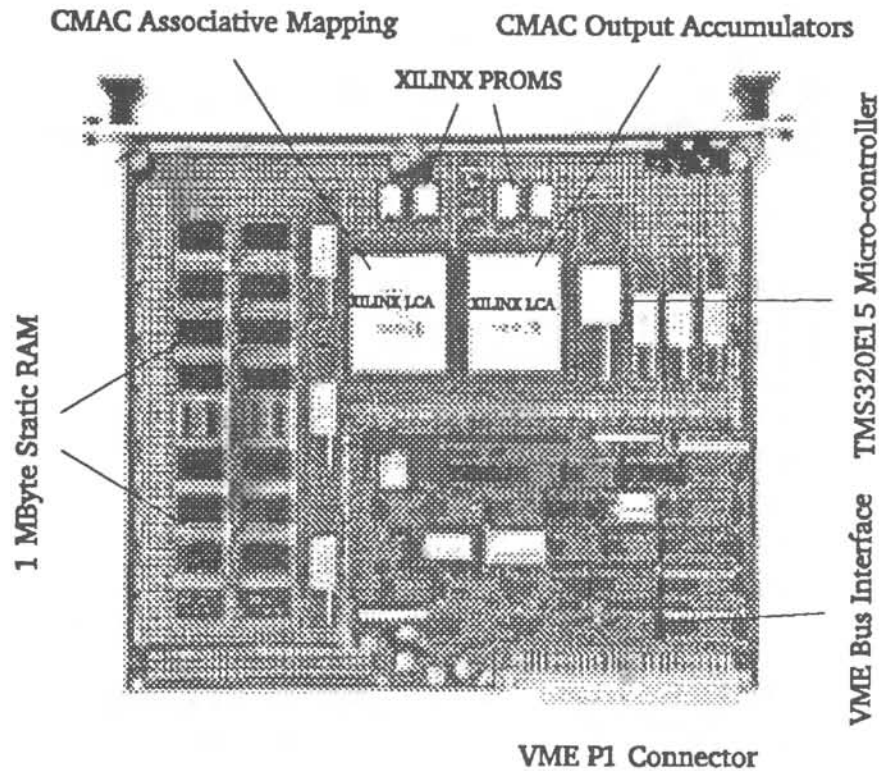

Figure 2: The Component Side of the VME Based CMAC Associative Memory Card. The two large XILINX 3090 logic cell arrays implement the CMAC associative mapping and the response accumulation/weight adjustment circuitry. The weights are stored in the 1 Mbyte static RAM. The TMS320E15 microcontroller supervises communications between the CMAC hardware and the VME host.

provide maximum flexibility in the logical organization of the CMAC associative memory, as viewed by the VME host system. The board can be initialized to act as from 1 to 8 independent virtual CMAC networks. For each network, the number of 16 bit inputs is selectable from 1 to 512, the number of 16 bit outputs is selectable from 1 to 8, and the number of overlapping receptive fields is selectable from 2 to 256.

Figure 3 shows typical response times during training and response generation operations for a CMAC network with 1 million adjustable weights. The data shown represent networks with 32 integer inputs and 8 integer outputs, with the number of overlapping receptive fields varied between 8 and 256. Throughout most of this range CMAC training and response times are well below 1 millisecond. These performance specifications should accommodate typical real time control problems (allowing 1000 cycle per second control rates), as well as many problems in pattern recognition.

A similar CMAC system for the 16 bit PC-AT bus has been developed by the Shenandoah Systems Company for commercial applications. This CMAC system supports both 8 and 16 bit adjustable weights (1 Mbyte total storage), and 8 independent virtual CMAC networks on a single card. Response times for the commercial CMAC-AT card are similar to those shown in Figure 3. A commercial version

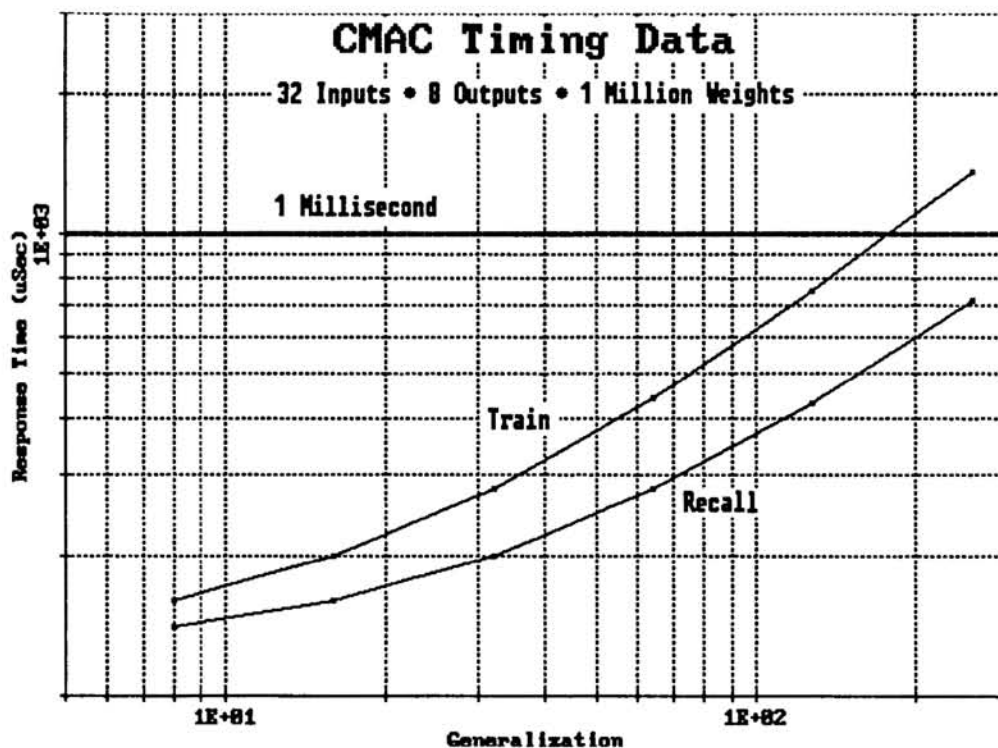

Figure 3: CMAC Associative Memory Response and Training Times. Response times are shown for values of the generalization parameter (the number of overlapping receptive fields) between 8 and 256. In each case the CMAC had 32 integer inputs, 8 integer outputs, and one million adjustable weights.

of the VME bus design is currently under development.

## Acknowledgements

This work was sponsored in part by the Office of Naval Research (ONR Grant Number N00014-89-J-1686) and the National Institute of Standards and Technology.

## References

Albus, J. S., *Theoretical and Experimental Aspects of a Cerebellar Model.* PhD Thesis, University of Maryland, Dec. 1972.

Albus, J. S., A New Approach to Manipulator Control: The Cerebellar Model Articulation Controller (CMAC). *Trans. of the ASME, Journal of Dynamic Systems, Measurement and Control,* vol. 97, pp. 220-227, September, 1975.

Albus, J. S., Mechanisms of Planning and Problem Solving in the Brain. *Mathematical Biosciences,* vol. 45, pp. 247-293, August, 1979.

Miller, W. T., A Nonlinear Learning Controller for Robotic Manipulators. *Proc. of the SPIE: Intelligent Robots and Computer Vision,* vol 726, pp. 416-423, October, 1986.

Miller, W. T., Sensor Based Control of Robotic Manipulators Using A General Learning Algorithm. *IEEE J. of Robotics and Automation,* vol. RA-3, pp. 157-165, April, 1987.

Miller, W. T., Glanz, F. H., and Kraft, L. G., Application of a General Learning Algorithm to the Control of Robotic Manipulators. *The International Journal of Robotics Research,* vol. 6.2, pp. 84-98, Summer, 1987.

Miller, W.T., and Hewes, R.P., Real Time Experiments in Neural Network Based Learning Control During High Speed, Nonrepetitive Robot Operations. *Proceedings of the Third IEEE International Symposium on Intelligent Control,* Washington, D.C., August 24-26, 1988.

Miller, W, T., Real Time Application of Neural Networks for Sensor-Based Control of Robots with Vision. *IEEE Transactions on Systems, Man, and Cybernetics.* Special issue on Information Technology for Sensory-Based Robot Manipulators, vol. 19, pp. 825-831, 1989.

Miller, W. T., Hewes, R. P., Glanz, F. H., and Kraft, L. G., Real Time Dynamic Control of an Industrial Manipulator Using a Neural Network Based Learning Controller. *IEEE J. of Robotics and Automation* vol. 6, pp. 1-9, 1990.

Glanz, F. H., Miller, W. T., Shape Recognition Using a CMAC Based Learning System. *Proceedings SPIE: Intelligent Robots and Computer Vision,* Cambridge, Mass., Nov., 1987.

Herold, D. J., Miller, W. T., Kraft, L. G., and Glanz, F. H., Pattern Recognition Using a CMAC Based Learning System. *Proceedings SPIE: Automated Inspection and High Speed Vision Architectures II,* vol. 1004, pp. 84-90, 1988.

Glanz, F. H., and Miller, W. T., Deconvolution and Nonlinear Inverse Filtering Using a Neural Network. *Proc. ICASSP 89,* Glasgow, Scotland, May 23-26, 1989, vol. 4, pp. 2349-2352.
